# Increase information transfer rates in BCI by CSP extension to multi-class

**Guido Dornhege[1], Benjamin Blankertz[1], Gabriel Curio[2], Klaus-Robert Müller[1,3]**

[1]Fraunhofer FIRST.IDA, Kekuléstr. 7, 12489 Berlin, Germany

[2]Neurophysics Group, Dept. of Neurology, Klinikum Benjamin Franklin,
Freie Universität Berlin, Hindenburgdamm 30, 12203 Berlin, Germany

[3]University of Potsdam, August-Bebel-Str. 89, 14482 Potsdam, Germany

{dornhege,blanker,klaus}@first.fraunhofer.de,
curio@zedat.fu-berlin.de

## Abstract

Brain-Computer Interfaces (BCI) are an interesting emerging technology that is driven by the motivation to develop an effective communication interface translating human intentions into a control signal for devices like computers or neuroprostheses. If this can be done bypassing the usual human output pathways like peripheral nerves and muscles it can ultimately become a valuable tool for paralyzed patients. Most activity in BCI research is devoted to finding suitable features and algorithms to increase information transfer rates (ITRs). The present paper studies the implications of using more classes, e.g., left vs. right hand vs. foot, for operating a BCI. We contribute by (1) a theoretical study showing under some mild assumptions that it is practically not useful to employ more than three or four classes, (2) two extensions of the common spatial pattern (CSP) algorithm, one interestingly based on simultaneous diagonalization, and (3) controlled EEG experiments that underline our theoretical findings and show excellent improved ITRs.

## 1   Introduction

The goal of a Brain-Computer Interface (BCI) is to establish a communication channel for translating human intentions – reflected by suitable brain signals – into a control signal for, e.g., a computer application or a neuroprosthesis (cf. [1]). If the brain signal is measured non-invasively by an electroencephalogram (EEG), if short training and preparation times are feasible and if it is possible to achieve high information transfer rates (ITRs), this interface can become a useful tool for disabled patients or an interesting gadget in the context of computer games. Recently, some approaches have been presented (cf. [1, 2]) which are good candidates for successfully implementing such an interface.

In a BCI system a subject tries to convey her/his intentions by behaving according to well-defined paradigms, like imagination of specific movements. An effective discrimination of different brain states is important in order to implement a suitable system for human subjects. Therefore appropriate features have to be chosen by signal processing techniques according to the selected paradigm. These features are translated into a control signal,

either by simple threshold criteria (cf. [1]), or by machine learning techniques where the computer learns a decision function from some training data [1, 3, 4, 5, 6].

For non-invasive BCI systems that are based on discrimination of voluntarily induced brain states three approaches are characteristic. (1) The Tübingen Thought Translation Device (TTD) [7] enables subjects to learn self-regulation of slow cortical potentials (SCP), i.e., electrocortical positivity and negativity. After some training in experiments with vertical cursor movement as feedback navigated by the SCP from central scalp position, patients are able to generate binary decisions in a 4-6 second pace with an accuracy of up to 85 %. (2) Users of the Albany BCI system [8] are able to control a cursor movement by their oscillatory brain activity into one of two or four possible targets on the computer screen and to achieve over 90 % hit rates after adapting to the system during many feedback sessions with a selection rate of 4 to 5 seconds in the binary decision problem. And (3), based on event-related modulations of the pericentral $\mu$- and/or $\beta$-rhythms of sensorimotor cortices (with a focus on motor preparation and imagination) the Graz BCI system [9] obtains accuracies of over 96 % in a ternary classification task with a trial duration of 8 seconds by evaluation of adaptive auto-regressive models (AAR). Note that there are other BCI systems which rely on stimulus/response paradigms, e.g. P300, see [1] for an overview.

In [10] an approach called *Common Spatial Patterns* (CSP) was suggested for use in a BCI context. This algorithm extracts event-related desynchronization (ERD) effects, i.e., event-related attenuations in some frequency bands, e.g., $\mu/\beta$-rhythm. However, the CSP algorithm can be used more generally, e.g., in [11] a suitable modification to movement-related potentials was presented. Further in [12] a first multi-class extension of CSP is presented which is based on pairwise classification and voting. In this paper we present further ways to extend this approach to many classes and compare to prior work.

By extending a BCI system to many classes a gain in performance can be obtained since the ITR can increase even if the percentage of correct classifications decreases. In [13] a first study for increasing the number of classes is demonstrated based on a hidden markov model approach. The authors conclude to use three classes which attains the highest ITR. We are focussing here on the same problem but using CSP extracted features and arrive at similar results. However, in a theoretical part we show that using more classes can be worth the effort if a suitable accuracy of all pairwise classifications is available. Consequently, extensions to multi-class settings are worthwhile for a BCI system, if and only if a suitable number of effectivly separable human brain states can be assigned.

## 2   How many brain states should be chosen?

Out of many different brain states (classes) our task is to find a subset of classes which is most profitable for the user of a BCI system. In this part we only focus on the information theoretical perspective. Using more classes holds the potential to increase ITR, although the rate of correct classifications decreases. For the subsequent theoretical considerations we assume gaussian distributions with equal covariance matrices for all classes which is a reasonable assumption for a wide range of EEG features, see section 4.3. Furthermore we assume equal priors between all classes. For three classes and equal pairwise classifications errors *err*, bounds for the expected classification error can be calculated in the following way: Let $(X,Y) \in \mathbb{R}^n \times \mathscr{Y}$ ($\mathscr{Y} = \{1,2,3\}$) be random variables and $P \sim \mathscr{N}(\mu_{1,2,3}, \Sigma)$ the probability distribution. Scaling appropriately we can assume $\Sigma = I$. We define the *optimal* classifier by $f^* : \mathbb{R}^n \to \mathscr{Y}$ with $f^* = argmin_{f \in F} P(f(X) \neq Y)$, where $F$ is some class of functions[1]. Similarly $f^*_{i,j}$ describes the optimal classifier between classes $i$ and $j$. Directly we get $err := P(f^*_{i,j}(X) \neq Y) = G(\|\mu_i - \mu_j\|/2)$ with $G(x) := \frac{1}{\sqrt{2\pi}} \int_x^\infty \exp(-x^2/2)dx$ and

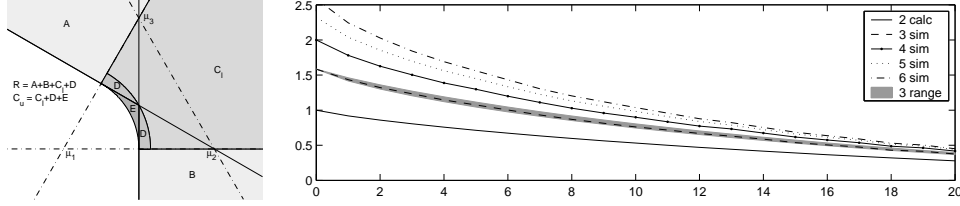

Figure 1: The figure on the left visualizes a method to estimate bounds for the ITR depending on the expected pairwise misclassification risk for three classes. The figure on the right shows the ITR [bits per decision] depending on the classification error [%] for simulated data for different number of classes (3-6 sim) and for 2 classes the real values (2 calc). Additionally the expected range (see (1)) (3 range) for three classes is visualized.

$i \neq j$. Therefore we get $\|\mu_j - \mu_i\|_2 = \Phi$ for all $i \neq j$ with some $\Phi > 0$ and finally due to symmetry and equal priors $P(f^*(X) \neq Y) = Q(\|X\|_2 \geq \min_{j=2,3}(\|X - \mu_j + \mu_1\|_2/2))$ where $Q \sim \mathcal{N}(0, I)$. Since evaluation of probabilities for polyhedrons in the gaussian space is hard, we only estimate lower and upper bounds. We can directly reduce the problem to a 2 dimensional space by shifting and rotating and by Fubini's theorem. Since $\|\mu_j - \mu_i\|_2 = \Phi$ for all $i \neq j$ the means lie at corners of an equilateral triangle (see Figure 1). We define $R := \{x \in \mathbb{R}^2 | \|x\|_2 \geq \|x - \mu_j + \mu_1\|_2, j = 2, 3\}$ and we can see after some calculation or by Figure 1 (left) with the sets defined there, that $A \cup B \cup C_l \subset R \subset A \cup B \cup C_u$. Due symmetry, the equilateral triangle and polar coordinates transformation we get finally

$$err + \frac{\exp(-\Phi^2/6)}{6} \quad \leq \quad P(f^*(X) \neq Y) \quad \leq \quad err + \frac{\exp(-\Phi^2/8)}{6}. \tag{1}$$

To compare classification performances involing different numbers of classes, we use the ITR quantified as bit rate per decision $I$ as defined due to Shannon's theorem: $I := \log_2 N + p \log_2(p) + (1-p) \log_2((1-p)/(N-1))$ per decision with number of classes $N$ and classification accuracy $p$ (cf. [14]). Figure 1 (right) shows the bounds in (1) for the ITR as a function of the expected pairwise misclassification errors. Additionally the same values on simulated data (100000 data points for each class) under the assumptions described above (equal pairwise performance, Gaussian distributed ...) are visualized for $N = 2, ..., 6$ classes. First of all, the figure confirms our estimated bounds. Furthermore the figure shows that under this strong assumptions extensions to multi-class are worthwhile. However, the gain of using more than 4 classes is tiny if the pairwise classification error is about 10 % or more. Under more realistic assumptions, i.e., more classes have increasing pairwise classification error compared to a wisely chosen subset it is improbable to increase the bit rate by increasing the number of classes higher than three or four. However, this depends strongly on the pairwise errors. If a suitable number of different brain states that can be discriminitated well, then indeed extensions to more classes are useful.

## 3 CSP and some multi-class extension

The CSP algorithm in its original form can be utilized for brain states that are characterized by a decrease or increase of a cortical rhythm with a characteristic topographic pattern.

### 3.1 CSP in a binary problem

Let $\Sigma_{1,2}$ be the centered covariance matrices calculated in the standard way of a trial-concatenated vector of dimension [channels $\times$ concatenated timepoints] belonging to the respective label. The computation of $\Sigma_{1,2}$ needs to be adapted to the paradigm, e.g., for slow cortical features such as the lateralized readiness potential (cf. [11]). The original CSP algorithm calculates a matrix $R$ and diagonal matrix $D$ with elements in $[0, 1]$ with

$$R\Sigma_1 R^T = D \qquad \text{and} \qquad R\Sigma_2 R^T = 1 - D \tag{2}$$

which can easily be obtained by whitening and spectral theory. Only a few projections with the highest ratio between their eigenvalues (lowest and highest ratios) are selected. Intuitively the CSP projections provide the scalp patterns which are most discriminative (see e.g. Figure 4).

## 3.2 Multi-Class extensions

*Using CSP within the classifier (IN):* This algorithm reduces a multi-class to several binary problems (cf. [15]) and was suggested in [12] for CSP in a BCI context. For all combinations of two different classes the CSP patterns are calculated as described in Eq.(2). The variances of the projections to CSP of every channel are used as input for an LDA-classifier for each 2-class combination. New trials are projected on these CSP patterns and are assigned to the class for which most classifiers are voting.

*One versus rest CSP (OVR):* We suggest a subtle modification of the approach above which permits to compute the CSP approach before the classification. We compute spatial patterns for each class against all others[2]. Then we project the EEG signals on all these CSP patterns, calculate the variances as before and then perform an LDA multi-class classification. The approach OVR appears rather similar to the approach IN, but there is in fact a large practical difference (additionally to the one-versus-rest strategy as opposed to pairwise binary subproblems). In the approach IN classification is only done binary on the CSP patterns according to the binary choice. OVR does multi-class classification on all projected signals.

*Simultaneous diagonalization (SIM):* The main trick in the binary case is that the CSP algorithm finds a simultaneous diagonalization of both covariance matrices whose eigenvalues sum to one. Thus a possible extension to many classes, i.e., many covariances $(\Sigma_i)_{i=1,...,N}$ is to find a matrix $R$ and diagonal matrices $(D_i)_{i=1,...N}$ with elements in $[0,1]$ and with $R\Sigma_i R^T = D_i$ for all $i = 1,...,N$ and $\sum_{i=1}^{N} D_i = I$. Such a decomposition can only be approximated for $N > 2$. There are several algorithms for approximate simultaneous diagonalization (cf. [16, 17]) and we are using the algorithm described in [18] due to its speed and reliability. As opposed to the two class problem there is no canonical way to choose the relevant CSP patterns. We explored several options such as using the highest or lowest eigenvalues. Finally, the best strategy was based on the assumption that two different eigenvalues for the same pattern have the same effect if their ratios to the mean of the eigenvalues of the other classes are multiplicatively inverse to each other, i.e., their product is 1. Thus all eigenvalues $\lambda$ are mapped to $\max(\lambda,(1-\lambda)/(1-\lambda+(N-1)^2\lambda))$ and a specified number $m$ of highest eigenvalues for each class are used as CSP patterns. It should be mentioned that each pattern is only used once, namely for the class which has the highest modified eigenvalue. If a second class would choose this pattern it is left out for this class and the next one is chosen. Finally variances are computed on the projected trials as before and conventional LDA multi-class classification is done.

## 4  Data acquisition and analysis methods

### 4.1  Experiments

We recorded brain activity from 4 subjects (codes *aa*, *af*, *ak* and *ar*) with multi-channel EEG amplifiers using 64 (128 for *aa*) channels band-pass filtered between 0.05 and 200 Hz and sampled at 1000 Hz. For offline analysis all signals were downsampled to 100 Hz. Surface EMG at both forearms and one leg, as well as horizontal and vertical EOG signals, were recorded to check for muscle activation and eye movements, but no trial was rejected.

The subjects in this experiment were sitting in a comfortable chair with arms lying relaxed on the armrests. All 4.5 seconds one of 6 different letters was appearing on the computer screen for 3 seconds. During this period the subject should imagine one of 6 different actions according to the displayed letter: imagination of *l*eft or *r*ight hand or *f*oot movement, or imagination of a *v*isual, *a*uditory or *t*actile sensation. Subject *aa* took only part in an experiment with the 3 classes *l*, *r* and *f*. 200 (resp. 160 for *aa*) trials for each class were recorded.

The aim of classification in these experiments is to discriminate trials of different classes using the whole period of imagination. A further reasonable objective to detect a new brainstate as fast as possible was not an object of this particular study. Note that the classes *v*, *a* and *t* were originally not intended to be BCI paradigms. Rather, these experiments were included to explore multi-class single-trial detection for brain states related to different sensory modalities for which it can reasonably be assumed that the regional activations can be well differentiated at a macroscopic scale of several centimeters.

## 4.2   Feature Extraction

Due to the fact that we focus on desynchronization effects (here the $\mu$-rhythm) we apply first a causal frequency filter of 8–15 Hz to the signals. Further, each trial consists of a two second window starting 500 ms after the visual stimulus. Then, the CSP algorithm is applied and finally variances of the projected trials were calculated to acquire the feature vectors. Alternatively, to see how effective the CSP algorithm is, the projection is left out for the binary classification task and we use instead techniques like Laplace filtering or common average reference (CAR) with a regularized LDA classifier on the variances.

The frequency band and the time period should be chosen individually by closer analysis of each data set. However, we are not focussing on this effect here, therefore we choose a setting which works well for all subjects. The number of chosen CSP patterns is a further variable. Extended search for different values can be done, but is omitted here. To have similar number of patterns for each algorithm we choose for IN 2 patterns from each side in each pairwise classification (resulting in $2N(N-1)$ patterns), for OVR 2 patterns from each side in each one-versus rest choice and for SIM 4 patterns for each class (both resulting in $4N$ patterns).

## 4.3   Classification and Validation

According to our studies the assumption that the features we are using are Gaussian distributed with equal covariance matrices holds well [2]. In this case Linear Discriminant Analysis (LDA) is optimal for classification in the sense that it minimizes the risk of misclassifications. Due to the low dimensionality of the CSP features regularization is not required.

To assess the classification performance, the generalization error was estimated by $10\times10$-fold cross-validation. Since the CSP algorithm depends on the class labels, the calculation of this projection is done in the cross-validation on each training set. Doing it on the whole data set beforehand can result in overfitting, i.e., underestimating the generalization error.

For the purpose of this paper the best configuration of classes should be found. The most sophisticated way in BCI context would have consisted in doing many experiment with different sets of classes. Unfortunately this is very time consuming and not of interest for the BCI user. A more useful way is to do in a preliminary step experiments with many classes and choose within an offline analysis which is the best subset by testing all combinations. With the best chosen class configuration the experiment should be repeated to confirm the results. However, in this paper we present results of this simpler experiment, in fact following the setting in [13].

# 5 Results

In Figure 2 the bit rates for all binary combinations of two classes and for all subjects are shown. The results for the CSP algorithm are contrasted in the plot with the results of LAPLACE/CAR in such a way that for points below the diagonal CSP is better and for points above the other algorithms are better. We can conclude that it is usually advantageous to use the CSP algorithm. Furthermore it is observable that the pairwise classification performances differ strongly. According to our theoretical considerations we should therefore assume that in the multi-class case a configuration with 3 classes will perform best.

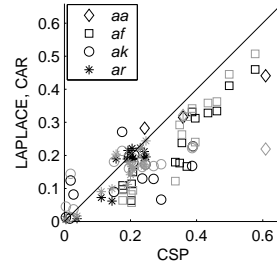

Figure 2: In the scatter plot the ITRs [bits per decision] for all 2-class combinations for all subjects obtained by CSP are shown on the *x*-axis while those by LAPLACE (dark points) resp. CAR (light points) are on the *y*-axis. That means for marks below the diagonal CSP outperforms LAPLACE resp. CAR.

Figure 3 shows the ITRs for all multi-class configurations (N=3,...,6) for different subjects. Results for baseline method IN are compared to the new methods SIM and OVR. The latter methods are superior for those configurations whose results are below the diagonal in the scatter plot. For an overview the upper plots show histograms of the *differences* in ITR between SIM/OVR and IN and a gaussian approximation. We can conclude from these figures that no algorithm is generally the best. SIM shows the best mean performance for subjects *ak* and *ar* but the performance falls off for subject *af*. Since for *aa* only one three class combination is available, we omit a visualization. However, SIM performs again best for this subject.

Statistical tests of significance are omitted since the classification results are generally not independent, e.g., classification of {*l,r,f*} and {*l,a,t*} are dependent since the trials of class *l* are involved in both. For a given number of classes Figure 4 shows the ITR obtained for the optimal subset of brain states by the best of the presented algorithms. As conjectured from fluctuations in pairwise discriminability, the bit rates decrease when using more than three classes. In three out of four subjects the peak ITR is obtained with three classes, only for subject *aa* pairwise classification is better. Here one further strategy is helpful. Additionally to the variance, autoregressive parameters can be calculated on the projections on the CSP patterns filtered here at 7–30 Hz and used for classification. In this case the pairwise classification errors are more balanced such that we acquire finally an ITR of 0.76

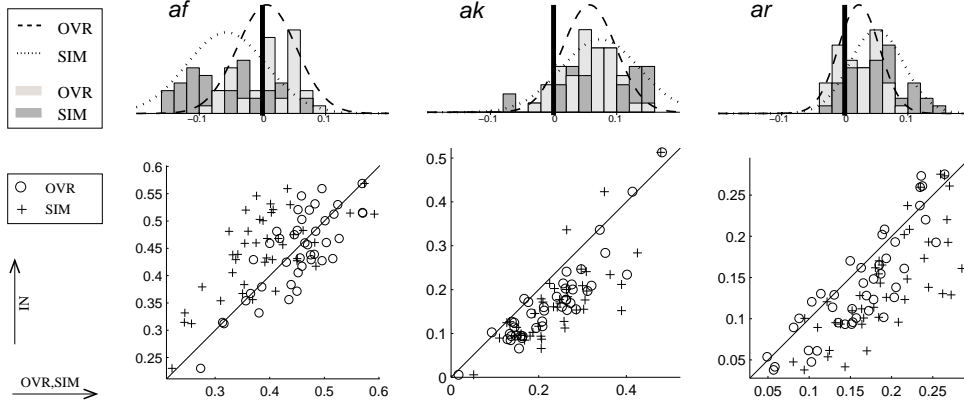

Figure 3: In the scatter plot the ITRs [bits per decision] obtained by the baseline method IN are shown on the *y*-axis while those by SIM (+) and OVR (○) are on the *x*-axis. That means for marks below the diagonal SIM resp. OVR outperforms IN. For an overall overview the upper plots show histograms of the differences in ITR between SIM/OVR and IN and shows a gaussian approximation of them. Here positive values belong to good performances of SIM and OVR.

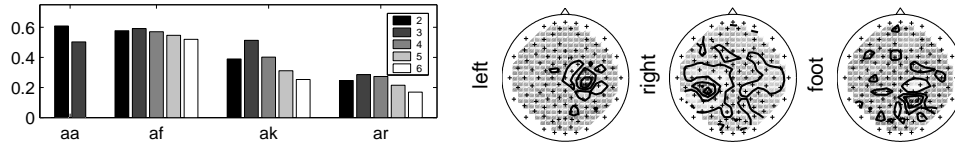

Figure 4: The figure on the left shows the ITR per trial for different number of classes with the best algorithm described above. The figure on the right visualizes the first pattern chosen by SIM for each class for *aa*.

per decision, whereas the best binary combination has 0.6 bits per decision. The worth of using AR for this subject are caused by different frequency bands in which discriminative informations are. For the other subjects similar gains could not be observed by using AR parameters.

Finally the CSP algorithm contains some further feature, namely that the spatial patterns can be plotted as scalp topographies. In Figure 4 the first pattern for each class of algorithm SIM is shown for subject *aa*. Evidently, this algorithm can reproduce neurophysiological prior knowledge about the location of ERD effects because for each activated limb the appropriate region of motor cortex is activated, e.g., a left (right) lateral site for the right (left) hand and an area closer to the central midline for the foot.

**Psychological perspective.** In principle, multi-class decisions can be derived from a decision space natural to human subjects. In a BCI context such set of decisions will be performed most 'intuitively', i.e., without a need for prolonged training, if the differential brain states are naturally related to a set of intended actions. This is the case, e.g., for movements of different body parts which have a somatotopically ordered lay-out in the primary motor cortex resulting in spatially discriminable patterns of EEG signals, such as readiness potentials or event-related desynchonizations specific for finger, elbow or shoulder movement intentions. In contrast, having to imagine a tune in order to move a cursor upwards vs imaging a visual scene to induce a downward movement will produce spatially discriminable patterns of EEG signals related to either auditory or visual imagery, but its action-effect-contingency would be counter-intuitive. While humans are able to adapt and to learn such complex tasks, this could take *weeks* of training before it would be performed fast, reliably and 'automatically'. Another important aspect of multi-class settings is that the use of more classes which is discriminated by the BCI device only at lower accuracy is likely to confuse the user.

## 6   Concluding discussion

Current BCI research strives for enhanced information transfer rates. Several options are available: (1) training of the BCI users, which can be somewhat tedious if up to 300 hours of training would be necessary, (2) invasive BCI techniques, which we consider not applicable for healthy human test subjects, (3) improved machine learning and signal processing methods where, e.g., new filtering, feature extraction and sophisticated classifiers are constantly tuned and improved[3], (4) faster trial speeds and finally (5) more classes among which the BCI user is choosing. This work analysed the theoretical and practical implications of using more than two classes, and also psychological issues were shortly discussed. In essence we found that higher a ITR is achieved with three classes, however, it seems unlikely that it can be increased by moving above four classes. This finding is confirmed in EEG experiments. As a further, more algorithmic, contribution we suggested two modifications of the CSP method for the multi-class case. As a side remark: our multi-class CSP algorithms also allow to gain a significant speed up in a real-time feedback experiment as filtering operations only need to be performed on very few CSP components (as opposed to on all channels). Since this corresponds to an implicit dimensionality reduction, good

results can be also achieved with CSP using less patterns/trials.

Comparing the results of SIM, OVR and IN we find that for most of the subjects SIM or OVR provide better results. Assuringly the algorithms SIM, OVR and IN allow to extract scalp pattern for the classification that match well with neurophysiological textbook knowledge (cf. Figure 4). In this paper the beneficial role of a third class was confirmed by an offline analysis. Future studies will therefore target on online experiments with more than two classes; first experimental results are promising. Another line of study will explore information from complementary neurophysiological effects in the spirit of [19] in combination with multi-class paradigms.

Finally it would be useful to explore configurations with more than two classes which are more natural and also more userfriendly from the psychological perspective discussed above.

**Acknowledgments** We thank S. Harmeling, M. Kawanabe, A. Ziehe, G. Rätsch, S. Mika, P. Laskov, D. Tax, M. Kirsch, C. Schäfer and T. Zander for helpful discussions. The studies were supported by BMBF-grants FKZ 01IBB02A and FKZ 01IBB02B.

## Footnotes

[1]For the moment we pay no attention to whether such a function exists. In the current set-up $F$ is usually the space of all linear classifiers, and under the probability assumptions mentioned above such a minimum exist.

[2]Note that this can be done similarly with pairwise patterns, but in our studies no substancial difference was observable and therefore one-versus-rest is favourable, since it chooses less patterns.

[3]See 1st and 2nd BCI competition: `http://ida.first.fraunhofer.de/~blanker/competition/`

## References

[1] J. R. Wolpaw, N. Birbaumer, D. J. McFarland, G. Pfurtscheller, and T. M. Vaughan, "Brain-computer interfaces for communication and control", *Clin. Neurophysiol.*, 113: 767–791, 2002.

[2] B. Blankertz, G. Dornhege, C. Schäfer, R. Krepki, J. Kohlmorgen, K.-R. Müller, V. Kunzmann, F. Losch, and G. Curio, "Boosting Bit Rates and Error Detection for the Classification of Fast-Paced Motor Commands Based on Single-Trial EEG Analysis", *IEEE Trans. Neural Sys. Rehab. Eng.*, 11(2): 127–131, 2003.

[3] B. Blankertz, G. Curio, and K.-R. Müller, "Classifying Single Trial EEG: Towards Brain Computer Interfacing", in: T. G. Diettrich, S. Becker, and Z. Ghahramani, eds., *Advances in Neural Inf. Proc. Systems (NIPS 01)*, vol. 14, 157–164, 2002.

[4] L. Trejo, K. Wheeler, C. Jorgensen, R. Rosipal, S. Clanton, B. Matthews, A. Hibbs, R. Matthews, and M. Krupka, "Multimodal Neuroelectric Interface Development", *IEEE Trans. Neural Sys. Rehab. Eng.*, 2003, accepted.

[5] L. Parra, C. Alvino, A. C. Tang, B. A. Pearlmutter, N. Yeung, A. Osman, and P. Sajda, "Linear spatial integration for single trial detection in encephalography", *NeuroImage*, 2002, to appear.

[6] W. D. Penny, S. J. Roberts, E. A. Curran, and M. J. Stokes, "EEG-Based Communication: A Pattern Recognition Approach", *IEEE Trans. Rehab. Eng.*, 8(2): 214–215, 2000.

[7] N. Birbaumer, N. Ghanayim, T. Hinterberger, I. Iversen, B. Kotchoubey, A. Kübler, J. Perelmouter, E. Taub, and H. Flor, "A spelling device for the paralysed", *Nature*, 398: 297–298, 1999.

[8] J. R. Wolpaw, D. J. McFarland, and T. M. Vaughan, "Brain-Computer Interface Research at the Wadsworth Center", *IEEE Trans. Rehab. Eng.*, 8(2): 222–226, 2000.

[9] B. O. Peters, G. Pfurtscheller, and H. Flyvbjerg, "Automatic Differentiation of Multichannel EEG Signals", *IEEE Trans. Biomed. Eng.*, 48(1): 111–116, 2001.

[10] H. Ramoser, J. Müller-Gerking, and G. Pfurtscheller, "Optimal spatial filtering of single trial EEG during imagined hand movement", *IEEE Trans. Rehab. Eng.*, 8(4): 441–446, 2000.

[11] G. Dornhege, B. Blankertz, and G. Curio, "Speeding up classification of multi-channel Brain-Computer Interfaces: Common spatial patterns for slow cortical potentials", in: *Proceedings of the 1st International IEEE EMBS Conference on Neural Engineering. Capri 2003*, 591–594, 2003.

[12] J. Müller-Gerking, G. Pfurtscheller, and H. Flyvbjerg, "Designing optimal spatial filters for single-trial EEG classification in a movement task", *Clin. Neurophysiol.*, 110: 787–798, 1999.

[13] B. Obermaier, C. Neuper, C. Guger, and G. Pfurtscheller, "Information Transfer Rate in a Five-Classes Brain-Computer Interface", *IEEE Trans. Neural Sys. Rehab. Eng.*, 9(3): 283–288, 2001.

[14] J. R. Wolpaw, N. Birbaumer, W. J. Heetderks, D. J. McFarland, P. H. Peckham, G. Schalk, E. Donchin, L. A. Quatrano, C. J. Robinson, and T. M. Vaughan, "Brain-Computer Interface Technology: A review of the First International Meeting", *IEEE Trans. Rehab. Eng.*, 8(2): 164–173, 2000.

[15] E. Allwein, R. Schapire, and Y. Singer, "Reducing multiclass to binary: A unifying approach for margin classifiers", *Journal of Machine Learning Research*, 1: 113–141, 2000.

[16] J.-F. Cardoso and A. Souloumiac, "Jacobi angles for simultaneous diagonalization", *SIAM J.Mat.Anal.Appl.*, 17(1): 161 ff., 1996.

[17] D.-T. Pham, "Joint Approximate Diagonalization of Positive Definite Matrices", *SIAM J. on Matrix Anal. and Appl.*, 22(4): 1136–1152, 2001.

[18] A. Ziehe, P. Laskov, K.-R. Müller, and G. Nolte, "A Linear Least-Squares Algorithm for Joint Diagonalization", in: *Proc. 4th International Symposium on Independent Component Analysis and Blind Signal Separation (ICA2003)*, 469–474, Nara, Japan, 2003.

[19] G. Dornhege, B. Blankertz, G. Curio, and K.-R. Müller, "Combining Features for BCI", in: S. Becker, S. Thrun, and K. Obermayer, eds., *Advances in Neural Inf. Proc. Systems (NIPS 02)*, vol. 15, MIT Press: Cambridge, MA, 2003.
